# Generalizable Relational Binding from Coarse-coded Distributed Representations

**Randall C. O'Reilly**
Department of Psychology
University of Colorado Boulder
345 UCB
Boulder, CO 80309
oreilly@psych.colorado.edu

**Richard S. Busby**
Department of Psychology
University of Colorado Boulder
345 UCB
Boulder, CO 80309
Richard.Busby@Colorado.EDU

## Abstract

We present a model of binding of relationship information in a spatial domain (e.g., square above triangle) that uses low-order *coarse-coded* conjunctive representations instead of more popular temporal synchrony mechanisms. Supporters of temporal synchrony argue that conjunctive representations lack both efficiency (i.e., combinatorial numbers of units are required) and systematicity (i.e., the resulting representations are overly specific and thus do not support generalization to novel exemplars). To counter these claims, we show that our model: a) uses far fewer hidden units than the number of conjunctions represented, by using coarse-coded, distributed representations where each unit has a broad tuning curve through high-dimensional conjunction space, and b) is capable of considerable generalization to novel inputs.

## 1 Introduction

The binding problem as it is classically conceived arises when different pieces of information are processed by entirely separate units. For example, we can imagine there are neurons that separately code for the shape and color of objects, and we are viewing a scene having a red triangle and a blue square (Figure 1). Because color and shape are encoded separately in this system, the internal representations do not discriminate this situation from one where we are viewing a red square and a blue triangle. This is the problem. Broadly speaking, there are two solutions to it. Perhaps the most popular solution is to imagine that binding is encoded by some kind of transient signal, such as temporal synchrony (e.g., von der Malsburg, 1981; Gray, Engel, Konig, & Singer, 1992; Hummel & Holyoak, 1997). Under this solution, the red and triangle units should fire together, as should the blue and square units, with each group firing out of phase with the other.

The other solution can be construed as solving the problem by questioning its fundamental assumption — that information is encoded completely separately in the first place (which is so seductive that it typically goes unnoticed). Instead, one can imagine that color and shape information are encoded *together* (i.e., *conjunctively*). In the red-triangle blue-square example, some neurons encode the conjunction of red and triangle, while others encode the conjunction of blue and square. Because these units are explicitly sensitive to these

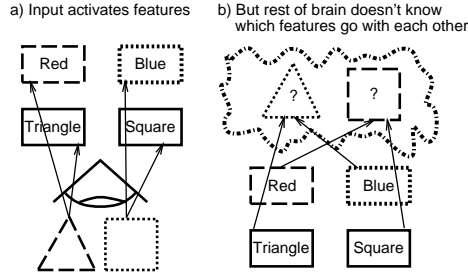

a) Input activates features  
b) But rest of brain doesn't know which features go with each other

Figure 1: Illustration of the binding problem. a) Visual inputs (red triangle, blue square) activate separate representations of color and shape properties. b) However, just the mere activation of these features does not distinguish for the rest of the brain the alternative scenario of a blue triangle and a red square. Red is indicated by dashed outline and blue by a dotted outline.

| obj1 | obj2 | R | G | B | S | C | T | RC GS BT | obj1 | obj2 | R | G | B | S | C | T | RC GS BT |
|------|------|---|---|---|---|---|---|----------|------|------|---|---|---|---|---|---|----------|
| RS | GC | 1 | 1 | 0 | 1 | 1 | 0 | 0 | RT | GC | 1 | 1 | 0 | 0 | 1 | 1 | 0 |
| RC | GS | 1 | 1 | 0 | 1 | 1 | 0 | 1 | RC | BT | 1 | 0 | 1 | 0 | 1 | 1 | 1 |
| RS | GT | 1 | 1 | 0 | 1 | 0 | 1 | 0 | RT | BC | 1 | 0 | 1 | 0 | 1 | 1 | 0 |
| RT | GS | 1 | 1 | 0 | 1 | 0 | 1 | 1 | GS | BC | 0 | 1 | 1 | 1 | 1 | 0 | 1 |
| RS | BC | 1 | 0 | 1 | 1 | 1 | 0 | 0 | GC | BS | 0 | 1 | 1 | 1 | 1 | 0 | 0 |
| RC | BS | 1 | 0 | 1 | 1 | 1 | 0 | 1 | GS | BT | 0 | 1 | 1 | 1 | 0 | 1 | 1 |
| RS | BT | 1 | 0 | 1 | 1 | 0 | 1 | 1 | GT | BS | 0 | 1 | 1 | 1 | 0 | 1 | 0 |
| RT | BS | 1 | 0 | 1 | 1 | 0 | 1 | 0 | GC | BT | 0 | 1 | 1 | 0 | 1 | 1 | 1 |
| RC | GT | 1 | 1 | 0 | 0 | 1 | 1 | 1 | GT | BC | 0 | 1 | 1 | 0 | 1 | 1 | 0 |

Table 1: Solution to the binding problem by using representations that encode combinations of input features (i.e., color and shape), but achieve greater efficiency by representing multiple such combinations. Obj1 and obj2 show the features of the two objects (R = Red, G = Green, B = Blue, S = Square, C = Circle, T = Triangle), and remaining columns show 6 localist units and one coarse-coded conjunctive unit. Adding this one conjunctive unit is enough to disambiguate the inputs.

conjunctions, they will not fire to a red square or a blue triangle, and thereby avoid the binding problem. The obvious problem with this solution, and one reason it has been largely rejected in the literature, is that it would appear to require far too many units to cover all of the possible conjunctions that need to be represented — a combinatorial explosion.

However, the combinatorial explosion problem is predicated on another seductive notion — that separate units are used for each possible conjunction. In short, both the binding problem itself and the problem with the conjunctive solution derive from *localist* assumptions about neural coding. In contrast, these problems can be greatly reduced by simply thinking in terms of *distributed* representations, where each unit encodes some possibly-difficult to describe amalgam of input features, such that individual units are active at different levels for different inputs, and many such units are active for each input (Hinton, McClelland, & Rumelhart, 1986). Therefore, the input is represented by a complex distributed pattern of activation over units, and each unit can exhibit varying levels of sensitivity to the featural conjunctions present in the input. The binding problem is largely avoided because a different pattern of activation will be present for a red-triangle, blue-square input as compared to a red-square, blue-triangle input.

These kinds of distributed representations can be difficult to understand. This is probably a significant reason why the ability of distributed representations to resolve the binding problem goes under-appreciated. However, we can analyze special cases of these representations to gain some insight. One such special case is shown in Table 1 from O'Reilly and Munakata (2000). Here, we add one additional distributed unit to an otherwise localist

featural encoding like that shown in Figure 1. This unit has a coarse-coded conjunctive representation, meaning that instead of coding for a single conjunction, it codes for several possible conjunctions. The table shows that if this set of conjunctions is chosen wisely, this single unit can enable the distributed pattern of activation across all units to distinguish between any two possible combinations of stimulus inputs. A more realistic system will have a larger number of partially redundant coarse-coded conjunctive units that will not require such precise representations from each unit. A similar demonstration was recently provided by Mel and Fiser (2000) in an analysis of distributed, low-order conjunctive representations (resembling "Wickelfeatures"; Wickelgren, 1969; Seidenberg & McClelland, 1989) in the domain of textual inputs. However, they did not demonstrate that a neural network learning mechanism would develop these representations, or that they could support systematic generalization to novel inputs.

## 2 Learning Generalizable Relational Bindings

We present here a series of models that test the ability of existing neural network learning mechanisms to develop low-order coarse-coded conjunctive representations in a challenging binding domain. Specifically, we focus on the problem of *relational* binding, which provides a link to higher-level cognitive function, and speaks to the continued use of *structured* representations in these domains. Furthermore, we conduct a critical test of these models in assessing their ability to generalize to novel inputs after moderate amounts of training. This is important because conjunctive representations might appear to limit generalization as these representations are more specific than purely localist representations. Indeed the inability to generalize is considered by some the primary limitation of conjunctive binding mechanisms (Holyoak & Hummel, 2000).

### 2.1 Relational Binding, Structured Representations, and Higher-level Cognition

A number of existing models rely on structured representations because they are regarded as essential for encoding complex relational information and other kinds of data structures that are used in symbolic models (e.g., lists, trees, sequences) (e.g., Touretzky, 1986; Shastri & Ajjanagadde, 1993; Hummel & Holyoak, 1997). A canonical example of a structured representation is a propositional encoding (e.g., *LIKES cats milk*) that has a main relational term (LIKES) that operates on a set of slot-like arguments that specify the items entering into the relationship. The primary advantages of such a representation are that it is transparently systematic or productive (anything can be put in the slots), and it is typically easy to compose more elaborate structures from these individual propositions (e.g., this proposition can have other propositions in its slots instead of just basic symbols).

The fundamental problem with structured representations, regardless of what implements them, is that they cannot be easily learned. To date, there have been no structured representation models that exhibit powerful learning of the form typically associated with neural networks. There are good reasons to believe that this reflects basic tradeoffs between complex structured representations and powerful learning mechanisms (Elman, 1991; St John & McClelland, 1990; O'Reilly & Munakata, 2000). Essentially, structured representations are discrete and fragile, and therefore do not admit to gradual changes over learning. In contrast, neural networks employ massively-parallel, graded processing that can search out many possible solutions at the same time, and optimize those that seem to make graded improvements in performance. In contrast, the discrete character of structured representations requires exhaustive combinatorial search in high-dimensional spaces.

To provide an alternative to these structured representations, our models test a simple example of relational encoding, focusing on easily-visualized spatial relationships, which can be thought of in propositional terms as for example *(LEFT-OF square triangle)*.

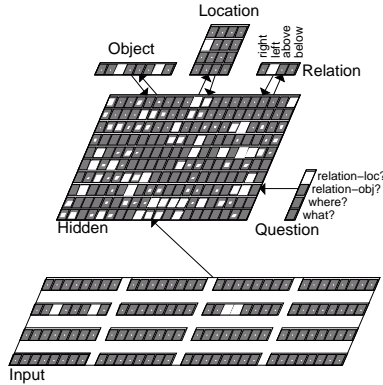

Figure 2: Spatial relationship binding model. Objects are represented by distributed patterns of activation over 8 features per location within a 4x4 array of locations. Inputs have two objects, arranged vertically or horizontally. The network answers questions posed by the Question input ("what", "where", and "what relationship?") —the answers require binding of object, location, and relationship information.

## 3 Spatial Relationship Binding Model

The spatial relationship binding model is shown in Figure 2. The overall framework for training the network is to present it with input patterns containing objects in different locations, and ask it various questions about these input displays. These questions ask about the identity and location of objects (i.e., "what?" and "where?"), and the relationships between the two objects (e.g., "where is object1 relative to object2?"). To answer these questions correctly, the hidden layer must bind object, location, and relationship information accurately in the hidden layer. Otherwise, it will confuse the two objects and their locations and relationships. Furthermore, we encoded the objects using distributed representations over features, so these features must be correctly bound into the same object.

Specifically, objects are represented by distributed patterns of activation over 8 features per location, in a 4x4 location array. Inputs have two different objects, arranged either vertically or horizontally. The network answers different questions about the objects posed by the Question input. For the "what?" question, the location of one of the objects is activated as an input in the Location layer, and the network must produce the correct object features for the object in that location. We also refer to this *target* object as the *agent* object. For the "where?" question, the object features for the agent object are activated in the Object layer, and the network must produce the correct location activation for that object. For the "relation-obj?" question, the object features for the agent object are activated, and the network must activate the relationship between this object and the other object (referred to as the *patient* object), in addition to activating the location for the agent object. For the "relation-loc?" question, the location of the agent object is activated, and the network must activate the relationship between this object and the patient object, in addition to activating the object features for the agent object.

This network architecture has a number of nice properties. For example, it has only one object and location encoding layer, both of which can act as either an input or an output. This is better than an alternative architecture having separate slots representing the agent and patient objects, because such slot-based encodings solve the binding problem by having separate role-specific units, which becomes implausible as the number of different roles and objects multiply. Note that supporting the dual input/output roles requires an interactive (recurrent, bidirectionally-connected) network (O'Reilly, 2001, 1998).

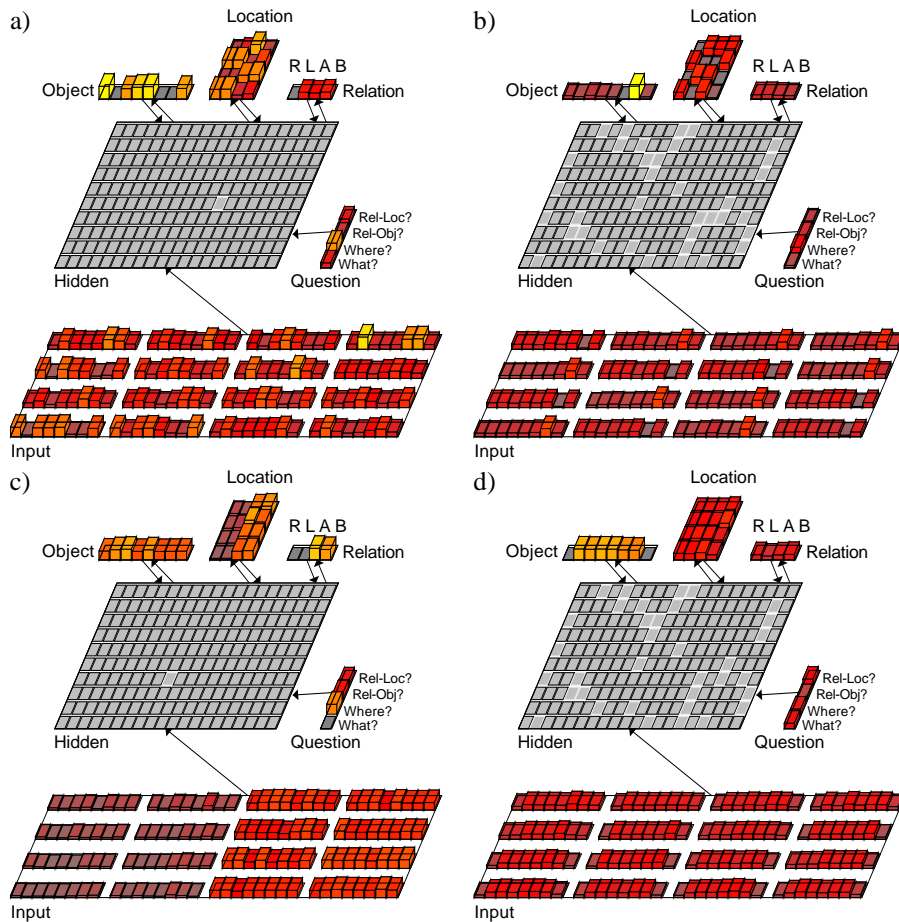

Figure 3: Hidden unit representations (values are weights into a hidden unit from all other layers) showing units (a & b) that bind object, location, & relationship information via low-order conjunctions, and other units that have systematic representations of location (c) and object features (d).

There are four levels of questions we can ask about this network. First, we can ask if standard neural network learning mechanisms are capable of solving this challenging binding problem. They are. Second, we can ask whether the network actually develops coarse-coded distributed representations. It does. Third, we can ask if these networks can generalize to novel inputs (both novel objects and novel locations for existing objects). They can. Finally, we can ask whether there are differences in how well different kinds of learning algorithms generalize, specifically comparing the *Leabra* algorithm with purely error-driven networks, as was recently done in other generalization tests with interactive networks (O'Reilly, 2001). This paper showed that interactive networks generalize significantly worse than comparable feedforward networks, but that good generalization can be achieved by adding additional biases or constraints on the learning mechanisms in the form of inhibitory competition and Hebbian learning in the Leabra algorithm. These results are replicated here, with Leabra generalization being roughly twice as good as other interactive algorithms.

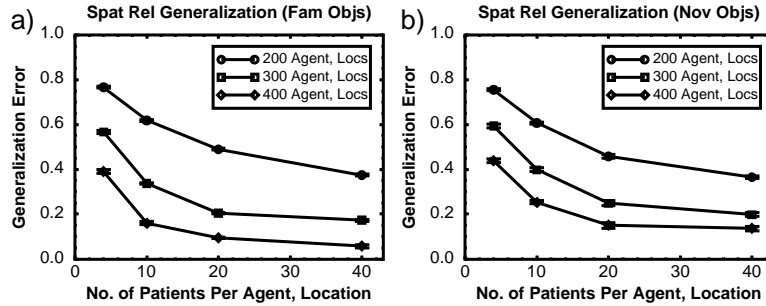

Figure 4: Generalization results (proportion errors on testing set) for the spatial relationship binding model using the Leabra algorithm as a function of the number of training items, specified as number of agent, location combinations and number of patient, locations per each agent, location. a) shows results for testing on familiar objects in novel locations. b) shows results for testing on novel objects that were never trained before.

## 3.1   Detailed Results

First, we examined the representations that developed in the network's hidden layer (Figure 3). Many units encoded low-order combinations (conjunctions) of object, location, and relationship features (Figure 3a & b). This is consistent with our hypothesis. Other units also encoded more systematic representations of location without respect to objects (Figure 3c) and object features without respect to location (Figure 3d).

To test the generalization capacity of the networks, we trained on only 26 of the 28 possible objects that can be composed out of 8 features with two units active, and only a subset of all 416 possible agent object x location combinations. We trained on 200, 300, and 400 such combinations. For each agent object-location input, there are 150 different patient object-location combinations per agent object-location, and we trained on 4, 10, 20, and 40, selected at random, for each different level of agent object-location combination training. At the most (400x40) there were a total of 16000 unique inputs trained out of a total possible of 62400, which amounts to about 1/4 of the training space. At the least (200x4) only roughly 1.3% of the training space was covered.

The ability of the network to generalize to the 26 familiar objects in novel locations was tested by measuring performance on a random sample of 640 of the untrained agent object-location combinations. The results for the Leabra algorithm are shown in Figure 4a. As one would expect, the number of training patterns improves generalization in a roughly proportional manner. Importantly, the network is able to generalize to a high level of performance, getting roughly 95% correct after training on only 25% of the training space (400x40), and achieving roughly 80% correct after training on only roughly 10% of the space (300x20).

The ability of the network to generalize to novel objects was tested by simply presenting the two novel objects as agents in all possible locations, with a random sampling of 20 different patients (which were the familiar objects), for a total of 640 different testing items (Figure 4b). Generalization on these novel objects was roughly comparable to the familiar objects, except there was an apparent ceiling point at roughly 15% generalization error where the generalization did not improve even with more training. Overall, the network performed remarkably well on these novel objects, and future work will explore generalization with fewer training objects.

To evaluate the extent to which the additional biologically-motivated biases in the Leabra algorithm are contributing to these generalization results, we ran networks using the contrastive Hebbian learning algorithm (CHL) and the Almeida-Pineda (AP) recurrent back-

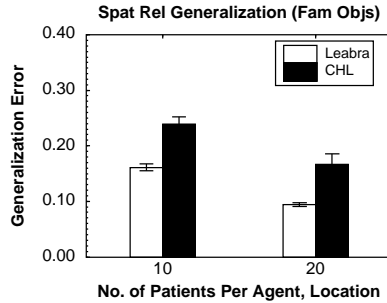

Figure 5: Generalization results for different algorithms on the spatial relationship binding task (see previous figure for details on measures) in the 400 x 10 or 20 conditions.

propagation algorithm, as in O'Reilly (2001). Both of these algorithms work in interactive, bidirectionally-connected networks, which are required for this task. Standard AP was unable to learn the task, we suspected because it does not preserve the symmetry of the weights as is required for stable settling. Attempts to to rectify this problem by enforcing symmetric weight changes did not succeed either. The results for CHL (Figure 5) replicated earlier results (O'Reilly, 2001) in showing that the additional biases in Leabra produced roughly twice as good of generalization performance compared to CHL.

## 4 Discussion

These networks demonstrate that existing, powerful neural network learning algorithms can learn representations that perform complex relational binding of information. Specifically, these networks had to bind together object identity, location, and relationship information to answer a number of questions about input displays containing two objects. This supports our contention that rich distributed representations containing coarse-coded conjunctive encodings can effectively perform binding. It is critical to appreciate that these distributed representations are highly efficient, encoding over 62400 unique input configurations with only 200 hidden units. Furthermore, these representations are *systematic*, in that they support generalization to novel inputs after training on a fraction of the input space.

Despite these initial successes, more work needs to be done to extend this approach to other kinds of domains that require binding. One early example of such an application is the St John and McClelland (1990) *sentence gestalt* model, which was able to sequentially process words in a sentence and construct a distributed internal representation of the meaning of the sentence (the sentence gestalt). This model was limited in that it required extremely large numbers of training trials and an elaborate training control mechanism. However, these limitations were eliminated in a recent replication of this model based on the Leabra algorithm (O'Reilly & Munakata, 2000). We plan to extend this model to handle a more complex corpus of sentences to more fully push the relational binding capacities of the model.

Finally, it is important to emphasize that we do not think that these low-order conjunctive representations are entirely sufficient to resolve the binding problems that arise in the cortex. One important additional mechanism is the use of selective attention to focus neural processing on coherent subsets of information present in the input (e.g., on individual objects, people, or conversations). The interaction between such a selective attentional system and a complex object recognition system was modeled in O'Reilly and Munakata (2000). In this model, selective attention was an emergent process deriving from excitatory interac-

tions between a spatial processing pathway and the object processing pathway, combined with surround inhibition as implemented by inhibitory interneurons. The resulting model was capable of sequentially processing individual objects when multiple such objects were simultaneously present in the input.

## Acknowledgments

This work was supported by ONR grant N00014-00-1-0246 and NSF grant IBN-9873492.

## 5    References

Elman, J. L. (1991). Distributed representations, simple recurrent networks, and grammatical structure. *Machine Learning*, *7*, 195–225.

Gray, C. M., Engel, A. K., Konig, P., & Singer, W. (1992). Synchronization of oscillatory neuronal responses in cat striate cortex —temporal properties. *Visual Neuroscience*, *8*, 337–347.

Hinton, G. E., McClelland, J. L., & Rumelhart, D. E. (1986). Distributed representations. In D. E. Rumelhart, J. L. McClelland, & PDP Research Group (Eds.), *Parallel distributed processing. Volume 1: Foundations* (Chap. 3, pp. 77–109). Cambridge, MA: MIT Press.

Holyoak, K. J., & Hummel, J. E. (2000). The proper treatment of symbols in a connectionist architecture. In E. Dietrich, & A. Markman (Eds.), *Cognitive dynamics: Conceptual and representational change in humans and machines*. Mahwah, NJ: Lawrence Erlbaum Associates.

Hummel, J. E., & Holyoak, K. J. (1997). Distributed representations of structure: A theory of analogical access and mapping. *Psychological Review*, *104*(3), 427–466.

Mel, B. A., & Fiser, J. (2000). Minimizing binding errors using learned conjunctive features. *Neural Computation*, *12*, 731–762.

O'Reilly, R. C. (1998). Six principles for biologically-based computational models of cortical cognition. *Trends in Cognitive Sciences*, *2*(11), 455–462.

O'Reilly, R. C. (2001). Generalization in interactive networks: The benefits of inhibitory competition and Hebbian learning. *Neural Computation*, *13*, 1199–1242.

O'Reilly, R. C., & Munakata, Y. (2000). *Computational explorations in cognitive neuroscience: Understanding the mind by simulating the brain*. Cambridge, MA: MIT Press.

Seidenberg, M. S., & McClelland, J. L. (1989). A distributed, developmental model of word recognition and naming. *Psychological Review*, *96*, 523–568.

Shastri, L., & Ajjanagadde, V. (1993). From simple associations to systematic reasoning: A connectionist representation of rules, variables, and dynamic bindings using temporal synchrony. *Behavioral and Brain Sciences*, *16*, 417–494.

St John, M. F., & McClelland, J. L. (1990). Learning and applying contextual constraints in sentence comprehension. *Artificial Intelligence*, *46*, 217–257.

Touretzky, D. S. (1986). BoltzCONS: Reconciling connectionism with the recursive nature of stacks and trees. *Proceedings of the 8th Annual Conference of the Cognitive Science Society* (pp. 522–530). Hillsdale, NJ: Lawrence Erlbaum Associates.

von der Malsburg, C. (1981). The correlation theory of brain function. MPI Biophysical Chemistry, Internal Report 81-2. In E. Domany, J. L. van Hemmen, & K. Schulten (Eds.), *Models of neural networks, II (1994)*. Berlin: Springer.

Wickelgren, W. A. (1969). Context-sensitive coding, associative memory, and serial order in (speech) behavior. *Psychological Review*, *76*, 1–15.
